# Efficient Recovery of Jointly Sparse Vectors

**Liang Sun, Jun Liu, Jianhui Chen, Jieping Ye**
School of Computing, Informatics, and Decision Systems Engineering
Arizona State University
Tempe, AZ 85287
{sun.liang,j.liu,jianhui.chen,jieping.ye}asu.edu

## Abstract

We consider the reconstruction of sparse signals in the multiple measurement vector (MMV) model, in which the signal, represented as a matrix, consists of a set of jointly sparse vectors. MMV is an extension of the single measurement vector (SMV) model employed in standard compressive sensing (CS). Recent theoretical studies focus on the convex relaxation of the MMV problem based on the $(2,1)$-norm minimization, which is an extension of the well-known 1-norm minimization employed in SMV. However, the resulting convex optimization problem in MMV is significantly much more difficult to solve than the one in SMV. Existing algorithms reformulate it as a second-order cone programming (SOCP) or semidefinite programming (SDP) problem, which is computationally expensive to solve for problems of moderate size. In this paper, we propose a new (dual) reformulation of the convex optimization problem in MMV and develop an efficient algorithm based on the prox-method. Interestingly, our theoretical analysis reveals the close connection between the proposed reformulation and multiple kernel learning. Our simulation studies demonstrate the scalability of the proposed algorithm.

## 1 Introduction

Compressive sensing (CS), also known as compressive sampling, has recently received increasing attention in many areas of science and engineering [3]. In CS, an unknown sparse signal is reconstructed from a single measurement vector. Recent theoretical studies show that one can recover certain sparse signals from far fewer samples or measurements than traditional methods [4, 8]. In this paper, we consider the problem of reconstructing sparse signals in the multiple measurement vector (MMV) model, in which the signal, represented as a matrix, consists of a set of jointly sparse vectors. MMV is an extension of the single measurement vector (SMV) model employed in standard compressive sensing.

The MMV model was motivated by the need to solve the neuromagnetic inverse problem that arises in Magnetoencephalography (MEG), which is a modality for imaging the brain [7]. It arises from a variety of applications, such as DNA microarrays [11], equalization of sparse communication channels [6], echo cancellation [9], magenetoencephalography [12], computing sparse solutions to linear inverse problems [7], and source localization in sensor networks [17]. Unlike SMV, the signal in the MMV model is represented as a set of jointly sparse vectors sharing their common nonzeros occurring in a set of locations [5, 7]. It has been shown that the additional block-sparse structure can lead to improved performance in signal recovery [5, 10, 16, 21].

Several recovery algorithms have been proposed for the MMV model in the past [5, 7, 18, 24, 25]. Since the sparse representation problem is a combinatorial optimization problem and is in general NP-hard [5], the algorithms in [18, 25] employ the greedy strategy to recover the signal using an iterative scheme. One alternative is to relax it into a convex optimization problem, from which the

global optimal solution can be obtained. The most widely studied approach is the one based on the $(2, 1)$-norm minimization [5, 7, 10]. A similar relaxation technique (via the 1-norm minimization) is employed in the SMV model. Recent studies have shown that most of theoretical results on the convex relaxation of the SMV model can be extended to the MMV model [5], although further theoretical investigation is needed [26]. Unlike the SMV model where the 1-norm minimization can be solved efficiently, the resulting convex optimization problem in MMV is much more difficult to solve. Existing algorithms formulate it as a second-order cone programming (SOCP) or semdefinite programming (SDP) [16] problem, which can be solved using standard software packages such as SeDuMi [23]. However, for problems of moderate size, solving either SOCP or SDP is computationally expensive, which limits their use in practice.

In this paper, we derive a dual reformulation of the $(2, 1)$-norm minimization problem in MMV. More especially, we show that the $(2, 1)$-norm minimization problem can be reformulated as a min-max problem, which can be solved efficiently via the prox-method with a nearly dimension-independent convergence rate [19]. Compared with existing algorithms, our algorithm can scale to larger problems while achieving high accuracy. Interestingly, our theoretical analysis reveals the close relationship between the resulting min-max problem and multiple kernel learning [14]. We have performed simulation studies and our results demonstrate the scalability of the proposed algorithm in comparison with existing algorithms.

**Notations:** All matrices are boldface uppercase. Vectors are boldface lowercase. Sets and spaces are denoted with calligraphic letters. The $p$-norm of the vector $\mathbf{v} = (v_1, \cdots, v_d)^T \in I\!\!R^d$ is defined as $\|\mathbf{v}\|_p := \left( \sum_{i=1}^d |v_i|^p \right)^{\frac{1}{p}}$. The inner product on $I\!\!R^{m \times d}$ is defined as $\langle \mathbf{X}, \mathbf{Y} \rangle = \text{tr}(\mathbf{X}^T \mathbf{Y})$. For matrix $\mathbf{A} \in I\!\!R^{m \times d}$, we denote by $\mathbf{a}^i$ and $\mathbf{a}_i$ the $i$th row and the $i$th column of $\mathbf{A}$, respectively. The $(r, p)$-norm of $\mathbf{A}$ is defined as:

$$\|\mathbf{A}\|_{r,p} := \left( \sum_{i=1}^m \|\mathbf{a}^i\|_r^p \right)^{\frac{1}{p}}. \tag{1}$$

## 2 The Multiple Measurement Vector Model

In the SMV model, one aims to recover the sparse signal $\mathbf{w}$ from a measurement vector $\mathbf{b} = \mathbf{A}\mathbf{w}$ for a given matrix $\mathbf{A}$ [3]. The SMV model can be extended to the multiple measurement vector (MMV) model, in which the signal is represented as a set of jointly sparse vectors sharing a common set of nonzeros [5, 7]. The MMV model aims to recover the sparse representations for SMVs simultaneously. It has been shown that the MMV model provably improves the standard CS recovery by exploiting the block-sparse structure [10, 21].

Specifically, in the MMV model we consider the reconstruction of the signal represented by a matrix $\mathbf{W} \in I\!\!R^{d \times n}$, which is given by a dictionary (or measurement matrix) $\mathbf{A} \in I\!\!R^{m \times d}$ and multiple measurement vector $\mathbf{B} \in I\!\!R^{m \times n}$ such that

$$\mathbf{B} = \mathbf{A}\mathbf{W}. \tag{2}$$

Each column of $\mathbf{A}$ is associated with an atom, and a set of atom is called a dictionary. A sparse representation means that the matrix $\mathbf{W}$ has a small number of rows containing nonzero entries. Usually, we have $m \ll d$ and $d > n$.

Similar to SMV, we can use $\|\mathbf{W}\|_{p,0}$ to measure the number of rows in $\mathbf{W}$ that contain nonzero entries. Thus, the problem of finding the sparsest representation of the signal $\mathbf{W}$ in MMV is equivalent to solving the following problem, a.k.a. the sparse representation problem:

$$(\text{P0}): \quad \min_{\mathbf{W}} \|\mathbf{W}\|_{p,0}, \quad \text{s.t.} \quad \mathbf{A}\mathbf{W} = \mathbf{B}. \tag{3}$$

Some typical choices of $p$ include $p = \infty$ and $p = 2$ [25]. However, solving (P0) requires enumerating all subsets of the set $\{1, 2, \cdots, d\}$, which is essentially a combinatorial optimization problem and is in general NP-hard [5]. Similar to the use of the 1-norm minimization in the SMV model, one natural alternative is to use $\|\mathbf{W}\|_{p,1}$ instead of $\|\mathbf{W}\|_{p,0}$, resulting in the following convex optimization problem (P1):

$$(\text{P1}): \quad \min_{\mathbf{W}} \|\mathbf{W}\|_{p,1}, \quad \text{s.t.} \quad \mathbf{A}\mathbf{W} = \mathbf{B}. \tag{4}$$

The relationship between (P0) and (P1) for the MMV model has been studied in [5].

For $p = 2$, the optimal $\mathbf{W}$ is given by solving the following convex optimization problem:

$$\min_{\mathbf{W}} \frac{1}{2}\|\mathbf{W}\|_{2,1}^2 \quad \text{s.t.} \quad \mathbf{AW} = \mathbf{B}. \tag{5}$$

Existing algorithms formulate Eq. (5) as a second-order cone programming (SOCP) problem or a semidefinite programming (SDP) problem [16]. Recall that the optimizaiton problem in Eq. (5) is equivalent to the following problem by removing the square in the objective:

$$\min_{\mathbf{W}} \frac{1}{2}\|\mathbf{W}\|_{2,1} \quad \text{s.t.} \quad \mathbf{AW} = \mathbf{B}.$$

By introducing auxiliary variable $t_i (i = 1, \cdots, d)$, this problem can be reformulated in the standard second-order cone programming (SOCP) formulation:

$$\min_{\mathbf{W},t_1,\cdots,t_d} \quad \frac{1}{2}\sum_{i=1}^{d} t_i \tag{6}$$
$$\text{s.t.} \quad \|\mathbf{W}^i\|_2 \leq t_i, t_i \geq 0, i = 1, \cdots, d, \quad \mathbf{AW} = \mathbf{B}.$$

Based on this SOCP formulation, it can also be transformed into the standard semidefinite programming (SDP) formulation:

$$\min_{\mathbf{W},t_1,\cdots,t_d} \quad \frac{1}{2}\sum_{i=1}^{d} t_i \tag{7}$$
$$\text{s.t.} \quad \begin{bmatrix} t_i\mathbf{I} & \mathbf{W}^{iT} \\ \mathbf{W}_i & t_i \end{bmatrix} \geq 0, t_i \geq 0, i = 1, \cdots, d, \quad \mathbf{AW} = \mathbf{B}.$$

The interior point method [20] and the bundle method [13] can be applied to solve SOCP and SDP. However, they do not scale to problems of moderate size, which limits their use in practice.

## 3 The Proposed Dual Formulation

In this section we present a dual reformulation of the optimization problem in Eq. (5). First, some preliminary results are summarized in Lemmas 1 and 2:

**Lemma 1.** *Let* $\mathbf{A}$ *and* $\mathbf{X}$ *be* $m$*-by-*$d$ *matrices. Then the following holds:*

$$\langle \mathbf{A}, \mathbf{X} \rangle \leq \frac{1}{2}\left(\|\mathbf{X}\|_{2,1}^2 + \|\mathbf{A}\|_{2,\infty}^2\right). \tag{8}$$

*When the equality holds, we have* $\|\mathbf{X}\|_{2,1} = \|\mathbf{A}\|_{2,\infty}$.

*Proof.* It follows from the definition of the $(r,p)$-norm in Eq. (1) that $\|\mathbf{X}\|_{2,1} = \sum_{i=1}^{m}\|\mathbf{x}^i\|_2$, and $\|\mathbf{A}\|_{2,\infty} = \max_{1 \leq i \leq m}\|\mathbf{a}^i\|_2$. Without loss of generality, we assume that $\|\mathbf{a}^k\|_2 = \max_{1 \leq i \leq m}\|\mathbf{a}^i\|_2$ for $1 \leq k \leq m$. Thus, $\|\mathbf{A}\|_{2,\infty} = \|\mathbf{a}^k\|_2$, and we have

$$\langle \mathbf{A}, \mathbf{X} \rangle = \sum_{i=1}^{m}\mathbf{a}^i\mathbf{x}^{iT} \leq \sum_{i=1}^{m}\|\mathbf{a}^i\|_2\|\mathbf{x}^i\|_2 \leq \sum_{i=1}^{m}\|\mathbf{a}^k\|_2\|\mathbf{x}^i\|_2 = \|\mathbf{a}^k\|_2\sum_{i=1}^{m}\|\mathbf{x}^i\|_2$$

$$\leq \frac{1}{2}\left(\|\mathbf{a}^k\|_2^2 + \left(\sum_{i=1}^{m}\|\mathbf{x}^i\|_2\right)^2\right) = \frac{1}{2}\left(\|\mathbf{A}\|_{2,\infty}^2 + \|\mathbf{X}\|_{2,1}^2\right).$$

Clearly, the last inequality becomes equality when $\|\mathbf{X}\|_{2,1} = \|\mathbf{A}\|_{2,\infty}$. $\qquad \square$

**Lemma 2.** *Let* $\mathbf{A}$ *and* $\mathbf{X}$ *be defined as in Lemma 1. Then the following holds:*

$$\max_{\mathbf{X}}\left\{\langle \mathbf{A}, \mathbf{X} \rangle - \frac{1}{2}\|\mathbf{X}\|_{2,1}^2\right\} = \frac{1}{2}\|\mathbf{A}\|_{2,\infty}^2.$$

*Proof.* Denote the set $\mathcal{Q} = \{k : 1 \le k \le m, \|\mathbf{a}^k\|_2 = \max_{1 \le i \le m} \|\mathbf{a}^i\|_2\}$. Let $\{\alpha_k\}_{k=1}^m$ be such that $\alpha_k = 0$ for $k \notin \mathcal{Q}$, $\alpha_k \ge 0$ for $k \in \mathcal{Q}$, and $\sum_{k=1}^m \alpha_k = 1$. Clearly, all inequalities in the proof of Lemma 1 become equalities if and only if we construct the matrix $\mathbf{X}$ as follows:

$$\mathbf{x}^k = \begin{cases} \alpha_k \mathbf{a}^k, & \text{if } k \in \mathcal{Q} \\ \mathbf{0}, & \text{otherwise.} \end{cases} \tag{9}$$

Thus, the maximum of $\langle \mathbf{A}, \mathbf{X} \rangle - \frac{1}{2}\|\mathbf{X}\|_{2,1}^2$ is $\frac{1}{2}\|\mathbf{A}\|_{2,\infty}^2$, which is achieved when $\mathbf{X}$ is constructed as in Eq. (9). $\square$

Based on the results established in Lemmas 1 and 2, we can derive the dual formulation of the optimization problem in Eq. (5) as follows. First we construct the Lagrangian $L$:

$$L(\mathbf{W}, \mathbf{U}) = \frac{1}{2}\|\mathbf{W}\|_{2,1}^2 - \langle \mathbf{U}, \mathbf{A}\mathbf{W} - \mathbf{B} \rangle = \frac{1}{2}\|\mathbf{W}\|_{2,1}^2 - \langle \mathbf{U}, \mathbf{A}\mathbf{W} \rangle + \langle \mathbf{U}, \mathbf{B} \rangle.$$

The dual problem can be formulated as follows:

$$\max_{\mathbf{U}} \min_{\mathbf{W}} \frac{1}{2}\|\mathbf{W}\|_{2,1}^2 - \langle \mathbf{U}, \mathbf{A}\mathbf{W} \rangle + \langle \mathbf{U}, \mathbf{B} \rangle. \tag{10}$$

It follows from Lemma 2 that

$$\min_{\mathbf{W}} \left\{ \frac{1}{2}\|\mathbf{W}\|_{2,1}^2 - \langle \mathbf{U}, \mathbf{A}\mathbf{W} \rangle \right\} = \min_{\mathbf{W}} \left\{ \frac{1}{2}\|\mathbf{W}\|_{2,1}^2 - \langle \mathbf{A}^T\mathbf{U}, \mathbf{W} \rangle \right\} = -\frac{1}{2}\|\mathbf{A}^T\mathbf{U}\|_{2,\infty}^2.$$

Note that from Lemma 2, the equality holds if and only if the optimal $\mathbf{W}^*$ can be represented as

$$\mathbf{W}^* = \text{diag}(\boldsymbol{\alpha})\mathbf{A}^T\mathbf{U}, \tag{11}$$

where $\boldsymbol{\alpha} = [\alpha_1, \cdots, \alpha_d]^T \in I\!\!R^d$, $\alpha_i \ge 0$ if $\|(\mathbf{A}^T\mathbf{U})^i\|_2 = \|\mathbf{A}^T\mathbf{U}\|_{2,\infty}$, $\alpha_i = 0$ if $\|(\mathbf{A}^T\mathbf{U})^i\|_2 < \|\mathbf{A}^T\mathbf{U}\|_{2,\infty}$, and $\sum_{i=1}^d \alpha_i = 1$. Thus, the dual problem can be simplified into the following form:

$$\max_{\mathbf{U}} -\frac{1}{2}\|\mathbf{A}^T\mathbf{U}\|_{2,\infty}^2 + \langle \mathbf{U}, \mathbf{B} \rangle. \tag{12}$$

Following the definition of the $(2, \infty)$-norm, we can reformulate the dual problem in Eq. (12) as a min-max problem, as summarized in the following theorem:

**Theorem 1.** *The optimization problem in Eq. (5) can be formulated equivalently as:*

$$\min_{\sum_{i=1}^d \theta_i = 1, \theta_i \ge 0} \max_{\mathbf{u}_1, \cdots, \mathbf{u}_n} \sum_{j=1}^n \left\{ \mathbf{u}_j^T \mathbf{b}_j - \frac{1}{2} \sum_{i=1}^d \theta_i \mathbf{u}_j^T \mathbf{G}_i \mathbf{u}_j \right\}, \tag{13}$$

*where the matrix $\mathbf{G}_i$ is defined as $\mathbf{G}_i = \mathbf{a}_i \mathbf{a}_i^T$ ($1 \le i \le d$), and $\mathbf{a}_i$ is the ith column of $\mathbf{A}$.*

*Proof.* Note that $\|\mathbf{A}^T\mathbf{U}\|_{2,\infty}^2$ can be reformulated as follows:

$$\|\mathbf{A}^T\mathbf{U}\|_{2,\infty}^2 = \max_{1 \le i \le d} \left\{ \|\mathbf{a}_i^T\mathbf{U}\|_2^2 \right\} = \max_{1 \le i \le d} \{ \text{tr}(\mathbf{U}^T \mathbf{a}_i \mathbf{a}_i^T \mathbf{U}) \} = \max_{1 \le i \le d} \{ \text{tr}(\mathbf{U}^T \mathbf{G}_i \mathbf{U}) \}$$

$$= \max_{\theta_i \ge 0, \sum_{i=1}^d \theta_i = 1} \sum_{i=1}^d \theta_i \text{tr}(\mathbf{U}^T \mathbf{G}_i \mathbf{U}). \tag{14}$$

Substituting Eq. (14) into Eq. (12), we obtain the following problem:

$$\max_{\mathbf{U}} -\frac{1}{2}\|\mathbf{A}^T\mathbf{U}\|_{2,\infty}^2 + \langle \mathbf{U}, \mathbf{B} \rangle \Leftrightarrow \max_{\mathbf{U}} \min_{\sum_{i=1}^d \theta_i = 1, \theta_i \ge 0} \langle \mathbf{U}, \mathbf{B} \rangle - \frac{1}{2}\sum_{i=1}^d \theta_i \text{tr}(\mathbf{U}^T \mathbf{G}_i \mathbf{U}). \tag{15}$$

Since the Slater's condition [2] is satisfied, the minimization and maximization in Eq. (15) can be exchanged, resulting in the min-max problem in Eq. (13). $\square$

**Corollary 1.** *Let $(\boldsymbol{\theta}^*, \mathbf{U}^*)$ be the optimal solution to Eq. (13) where $\boldsymbol{\theta}^* = (\theta_1^*, \cdots, \theta_d^*)^T$. If $\theta_i^* > 0$, then $\|(\mathbf{A}^T\mathbf{U}^*)^i\|_2 = \|\mathbf{A}^T\mathbf{U}^*\|_{2,\infty}$.*

Based on the solution to the dual problem in Eq. (13), we can construct the optimal solution to the primal problem in Eq. (5) as follows. Let $\mathbf{W}^*$ be the optimal solution of Eq. (5). It follows from Lemma 2 that we can construct $\mathbf{W}^*$ based on $\mathbf{A}^T\mathbf{U}^*$ as in Eq. (11). Recall that $\mathbf{W}^*$ must satisfy the equality constraint $\mathbf{AW}^* = \mathbf{B}$. The main result is summarized in the following theorem:

**Theorem 2.** *Given* $\mathbf{W}^* = diag(\boldsymbol{\alpha})\mathbf{A}^T\mathbf{U}^*$, *where* $\boldsymbol{\alpha} = [\alpha_1, \cdots, \alpha_d] \in I\!\!R^d$, $\alpha_i \geq 0$, $\alpha_i > 0$ *only if* $\|\left(\mathbf{A}^T\mathbf{U}^*\right)^i\|_2 = \|\mathbf{A}^T\mathbf{U}^*\|_{2,\infty}$, *and* $\sum_{i=1}^d \alpha_i = 1$. *Then,* $\mathbf{AW}^* = \mathbf{B}$ *if and only if* $(\boldsymbol{\alpha}, \mathbf{U}^*)$ *is the optimal solution to the problem in Eq. (13).*

*Proof.* First we assume that $(\boldsymbol{\alpha}, \mathbf{U}^*)$ is the optimal solution to the problem in Eq. (13). It follows that the partial derivative of the objective function with respect to $\mathbf{U}^*$ in Eq. (13) is 0, that is,

$$\mathbf{B} - \mathbf{A}\text{diag}(\boldsymbol{\alpha})\mathbf{A}^T\mathbf{U}^* = 0 \Leftrightarrow \mathbf{AW}^* = \mathbf{B}.$$

Next we prove the reverse direction by assuming $\mathbf{AW}^* = \mathbf{B}$. Since $\mathbf{W}^* = \text{diag}(\alpha)\mathbf{A}^T\mathbf{U}^*$, we have

$$0 = \mathbf{B} - \mathbf{AW}^* = \mathbf{B} - \mathbf{A}\text{diag}(\alpha_1, \cdots, \alpha_d)\mathbf{A}^T\mathbf{U}^*. \tag{16}$$

Define the function $\phi(\theta_1, \cdots, \theta_d, \mathbf{U})$ as

$$\phi(\theta_1, \cdots, \theta_d, \mathbf{U}) = \langle \mathbf{U}, \mathbf{B} \rangle - \frac{1}{2}\sum_{i=1}^d \theta_i \text{tr}(\mathbf{U}^T\mathbf{G}_i\mathbf{U}) = \sum_{j=1}^n \left\{ \mathbf{u}_j^T\mathbf{b}_j - \frac{1}{2}\sum_{i=1}^d \theta_i \mathbf{u}_j^T\mathbf{G}_i\mathbf{u}_j \right\}.$$

We consider the function $\phi(\alpha_1, \cdots, \alpha_d, \mathbf{U})$ with fixed $\theta_i = \alpha_i (1 \leq i \leq d)$. Note that this function is concave with respect to $\mathbf{U}$, thus its maximum is achieved when its partial derivative with respect to $\mathbf{U}$ is zero. It follows from Eq. (16) that $\frac{\partial \phi}{\partial \mathbf{U}}$ is zero when $\mathbf{U} = \mathbf{U}^*$. Thus, we have

$$\forall \mathbf{U}, \phi(\alpha_1, \cdots, \alpha_d, \mathbf{U}) \leq \phi(\alpha_1, \cdots, \alpha_d, \mathbf{U}^*).$$

With a fixed $\mathbf{U} = \mathbf{U}^*$, $\phi(\theta_1, \cdots, \theta_d, \mathbf{U}^*)$ is a linear combination of $\theta_i (1 \leq i \leq d)$ as:

$$\phi(\theta_1, \cdots, \theta_d, \mathbf{U}^*) = \langle \mathbf{U}^*, \mathbf{B} \rangle - \frac{1}{2}\sum_{i=1}^d \theta_i \|(\mathbf{A}^T\mathbf{U}^*)^i\|_2^2.$$

By the assumption, we have $\|(\mathbf{A}^T\mathbf{U}^*)^i\| = \|\mathbf{A}^T\mathbf{U}^*\|_{2,\infty}$, if $\alpha_i > 0$. Thus, we have

$$\phi(\alpha_1, \cdots, \alpha_d, \mathbf{U}^*) \leq \phi(\theta_1, \cdots, \theta_d, \mathbf{U}^*), \forall \theta_1, \cdots, \theta_d \text{ satisfying } \sum_{i=1}^d \theta_i = 1, \theta_i \geq 0.$$

Therefore, for any $\mathbf{U}, \theta_1, \cdots, \theta_d$ such that $\sum_{i=1}^d \theta_i = 1, \theta_i \geq 0$, we have

$$\phi(\alpha_1, \cdots, \alpha_d, \mathbf{U}) \leq \phi(\alpha_1, \cdots, \alpha_d, \mathbf{U}^*) \leq \phi(\theta_1, \cdots, \theta_d, \mathbf{U}^*), \tag{17}$$

which implies that $(\alpha_1, \cdots, \alpha_d, \mathbf{U}^*)$ is a saddle point of the min-max problem in Eq. (13). Thus, $(\boldsymbol{\alpha}, \mathbf{U}^*)$ is the optimal solution to the problem in Eq. (13). $\qquad\square$

Theorem 2 shows that we can reconstruct the solution to the primal problem based on the solution to the dual problem in Eq. (13). It paves the way for the efficient implementation based on the min-max formulation in Eq.(13). In this paper, the prox-method [19], which is discussed in detail in the next section, is employed to solve the dual problem in Eq. (13).

An interesting observation is that the resulting min-max problem in Eq. (13) is closely related to the optimization problem in multiple kernel learning (MKL) [14]. The min-max problem in Eq. (13) can be reformulated as

$$\min_{\sum_{i=1}^d \theta_i = 1, \theta_i \geq 0} \max_{\mathbf{u}_1, \cdots, \mathbf{u}_n} \sum_{j=1}^n \left\{ \mathbf{u}_j^T\mathbf{b}_j - \frac{1}{2}\mathbf{u}_j^T\mathbf{G}\mathbf{u}_j \right\}, \tag{18}$$

where the positive semidefinite (kernel) matrix $\mathbf{G}$ is constrained as a linear combination of a set of base kernels $\left\{ \mathbf{G}_i = \mathbf{a}^i\mathbf{a}^{iT} \right\}_{i=1}^d$ as $\mathbf{G} = \sum_{i=1}^d \theta_i\mathbf{G}_i$.

The formulation in Eq. (18) connects the MMV problem to MKL. Many efficient algorithms [14, 22, 27] have been developed in the past for MKL, which can be applied to solve (13). In [27], an extended level set method was proposed to solve MKL, which was shown to outperform the one based on the semi-infinite linear programming formulation [22]. However, the extended level set method involves a linear programming in each iteration and its theoretical convergence rate of $\mathcal{O}(1/\sqrt{N})$ ($N$ denotes the number of iterations) is slower than the proposed algorithm presented in the next section.

# 4 The Main Algorithm

We propose to employ the prox-method [19] to solve the min-max formulation in Eq. (13), which has a differentiable and convex-concave objective function. The algorithm is called "MMV$_{\text{prox}}$". The prox-method is a first-order method [1, 19] which is specialized for solving the saddle point problem and has a nearly dimension-independent convergence rate of $O(1/N)$ ($N$ denotes the number of iterations). We show that each iteration of MMV$_{\text{prox}}$ has a low computational cost, thus it scales to large-size problems.

The key idea is to convert the min-max problem to the associated variational inequality (v.i.) problem, which is then iteratively solved by a series of v.i. problems. Let $\mathbf{z} = (\boldsymbol{\theta}, \mathbf{U})$. The problem in Eq. (13) is equivalent to the following associated v.i. problem [19]:

$$\text{Find } \mathbf{z}^* = (\boldsymbol{\theta}^*, \mathbf{U}^*) \in \mathcal{S} : \langle F(\mathbf{z}^*), \mathbf{z} - \mathbf{z}^* \rangle \geq 0, \forall \mathbf{z} \in \mathcal{S}, \mathcal{S} = \mathcal{X} \times \mathcal{Y}, \tag{19}$$

where

$$F(\mathbf{z}) = \left( \frac{\partial}{\partial \boldsymbol{\theta}} \phi(\boldsymbol{\theta}, \mathbf{U}), -\frac{\partial}{\partial \mathbf{U}} \phi(\boldsymbol{\theta}, \mathbf{U}) \right) \tag{20}$$

is an operator constituted by the gradient of $\phi(\cdot, \cdot)$, $\mathcal{X} = \{\boldsymbol{\theta} \in I\!\!R^d : \|\theta\|_1 = 1, \theta_i \geq 0\}$, and $\mathcal{Y} = I\!\!R^{m \times n}$.

In solving the v.i. problem in Eq. (19), one key building block is the following projection problem:

$$P_{\mathbf{z}}(\bar{\mathbf{z}}) = \arg \min_{\tilde{\mathbf{z}} \in \mathcal{S}} \left[ \frac{1}{2} \|\tilde{\mathbf{z}}\|_2^2 + \langle \tilde{\mathbf{z}}, \bar{\mathbf{z}} - \mathbf{z} \rangle \right], \tag{21}$$

where $\bar{\mathbf{z}} = (\bar{\boldsymbol{\theta}}, \bar{\mathbf{U}})$ and $\tilde{\mathbf{z}} = (\tilde{\boldsymbol{\theta}}, \tilde{\mathbf{U}})$. Denote $(\boldsymbol{\theta}_*, \mathbf{U}_*) = P_{\mathbf{z}}(\bar{\mathbf{z}})$. It is easy to verify that

$$\boldsymbol{\theta}_* = \arg \min_{\tilde{\boldsymbol{\theta}} \in \mathcal{X}} \frac{1}{2} \|\tilde{\boldsymbol{\theta}} - (\boldsymbol{\theta} - \bar{\boldsymbol{\theta}})\|_2^2, \tag{22}$$

and

$$\mathbf{U}_* = \mathbf{U} - \bar{\mathbf{U}}. \tag{23}$$

Following [19], we present the pseudocode of the proposed MMV$_{\text{prox}}$ algorithm in Algorithm 1. In each iteration, we compute the projection (21) so that $\mathbf{w}_{t,s}$ is sufficiently close to $\mathbf{w}_{t,s-1}$ (controlled by the parameter $\delta$). It has been shown in [19] that, when $\gamma \leq \frac{1}{\sqrt{2}L}$ [$L$ denotes the Lipschitz continuous constant of the operator $F(\cdot)$], the inner iteration converges within two iterations, i.e., $\mathbf{w}_{t,2} = \mathbf{w}_{t,1}$ always holds. Moreover, Algorithm 1 has a global dimension-independent convergence rate of $O(1/N)$.

---

**Algorithm 1** The MMV$_{\text{prox}}$ Algorithm

---

**Input:** $\mathbf{A}, \mathbf{B}, \gamma, \mathbf{z}_0 = (\boldsymbol{\theta}_0, \mathbf{U}_0)$, and $\delta$
**Output:** $\boldsymbol{\theta}, \mathbf{U}$ and $\mathbf{W}$.
**Step $t$ ($t \geq 1$):** Set $\mathbf{w}_{t,0} = \mathbf{z}_{t-1}$ and find the smallest $s = 1, 2, \ldots$ such that

$$\mathbf{w}_{t,s} = P_{\mathbf{z}_{t-1}}(\gamma F(\mathbf{w}_{t,s-1})), \|\mathbf{w}_{t,s} - \mathbf{w}_{t,s-1}\|_2 \leq \delta.$$

Set $\mathbf{z}_t = \mathbf{w}_{t,s}$
**Final Step:** Set $\boldsymbol{\theta} = \frac{\sum_{i=1}^{t} \boldsymbol{\theta}_i}{t}, \mathbf{U} = \frac{\sum_{i=1}^{t} \mathbf{U}_i}{t}, \mathbf{W} = \text{diag}(\boldsymbol{\theta})\mathbf{A}^T\mathbf{U}.$

---

**Time Complexity** It costs $O(dmn)$ to evaluate the operator $F(\cdot)$ at a given point. $\boldsymbol{\theta}_*$ in Eq. (22) involves the Euclidean projection onto the simplex [1], which can be solved in linear time, i.e., in $O(d)$; and $\mathbf{U}_*$ in Eq. (23) can be analytically computed in $O(mn)$ time. Recall that at each iteration $t$, the inner iteration is at most 2. Thus, the time complexity for any given outer iteration is $O(dmn)$. Our analysis shows that MMV$_{\text{prox}}$ scales to large-size problems.

In comparison, the second-order methods such as SOCP have a much higher complexity per iteration. According to [15], the SOCP in Eq. (6) costs $O(d^3(n+1)^3)$ per iteration. In MMV, $d$ is typically larger than $m$. In this case, the proposed MMV$_{\text{prox}}$ algorithm has a much smaller cost per iteration than SOCP. This explains why MMV$_{\text{prox}}$ scales better than SOCP, as shown in our experiments in the next section.

Table 1: The averaged recovery results over 10 experiments ($d = 100$, $m = 50$, and $n = 80$).

| Data set | $\sqrt{\|\mathbf{W} - \mathbf{W}_p\|_F^2/(dn)}$ | $\sqrt{\|\mathbf{AW}_p - \mathbf{B}\|_F^2/(mn)}$ |
|---|---|---|
| 1 | 3.2723e-6 | 1.4467e-5 |
| 2 | 3.4576e-6 | 1.8234e-5 |
| 3 | 2.6971e-6 | 1.4464e-5 |
| 4 | 2.4099e-6 | 1.4460e-5 |
| 5 | 2.9611e-6 | 1.4463e-5 |
| 6 | 2.5701e-6 | 1.4459e-5 |
| 7 | 2.0884e-6 | 1.4469e-5 |
| 8 | 2.3454e-6 | 1.4475e-5 |
| 9 | 2.6807e-6 | 1.4461e-5 |
| 10 | 2.7172e-6 | 1.4481e-5 |
| Mean | 2.7200e-6 | 1.4843e-5 |
| Std | 4.1728e-7 | 1.1914e-6 |

## 5 Experiments

In this section, we conduct simulations to evaluate the proposed MMV$_{\text{prox}}$ algorithm in terms of the recovery quality and scalability.

**Experiment Setup** We generated a set of synthetic data sets (by varying the values of $m$, $n$, and $d$) for our experiments: the entries in $\mathbf{A} \in I\!R^{m \times d}$ were independently generated from the standard normal distribution $\mathcal{N}(0, 1)$; $\mathbf{W} \in I\!R^{d \times n}$ (the ground truth of the recovery problems) was generated in two steps: (1) randomly select $k$ rows with nonzero entries; (2) randomly generate the entries of those $k$ rows from $\mathcal{N}(0, 1)$. We denote by $\mathbf{W}_p$ the solution obtained from the proposed MMV$_{\text{prox}}$ algorithm. Ideally, $\mathbf{W}_p$ should be close to $\mathbf{W}$. Our experiments were performed on a PC with Intel Core 2 Duo T9500 2.6G CPU and 4G RAM. We employed the optimization package SeDuMi [23] for solving the SOCP formulation. All codes were implemented in Matlab. In all experiments, we terminate MMV$_{\text{prox}}$ when the change of the consecutive approximate solutions is less than 1e-6.

**Recovery Quality** In this experiment, we evaluate the recovery quality of the proposed MMV$_{\text{prox}}$ algorithm. We applied MMV$_{\text{prox}}$ on the data sets of size $d = 100$, $m = 50$, $n = 80$, and reported the averaged experimental results over 10 random repetitions. We measured the recovery quality in terms of the mean squared error: $\sqrt{\|\mathbf{W} - \mathbf{W}_p\|_F^2/(dn)}$. We also reported $\sqrt{\|\mathbf{AW}_p - \mathbf{B}\|_F^2/(mn)}$, which measures the violation of the constraint in Eq. (5). The experimental results are presented in Table 1. We can observe from the table that MMV$_{\text{prox}}$ recovers the sparse signal successfully in all cases.

Next, we study how the recovery error changes as the sparsity of $\mathbf{W}$ varies. Specifically, we applied MMV$_{\text{prox}}$ on the data sets of size $d = 100, m = 400$, and $n = 10$ with $k$ (the number of nonzero rows of $\mathbf{W}$) varying from $0.05d$ to $0.7d$, and used $\sqrt{\|\mathbf{W} - \mathbf{W}_p\|_F^2/(dn)}$ as the recovery quality measure. The averaged experimental results over 20 random repetitions are presented in Figure 1. We can observe from the figure that MMV$_{\text{prox}}$ works well in all cases, and a larger $k$ (less sparse $\mathbf{W}$) tends to result in a larger recovery error.

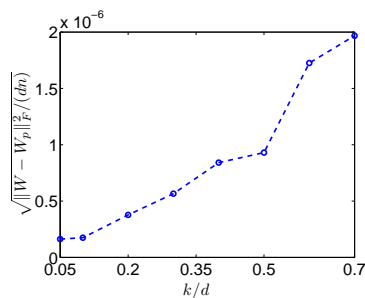

Figure 1: The increase of the recovery error as the sparsity level decreases

**Scalability** In this experiment, we study the scalability of the proposed MMV$_{prox}$ algorithm. We generated a collection of data sets by varying $m$ from 10 to 200 with a step size of 10, and setting $n = 2m$ and $d = 4m$ accordingly. We applied SOCP and MMV$_{prox}$ on the data sets and recorded their computation time. The experimental results are presented in Figure 2 (a), where the $x$-axis corresponds to the value of $m$, and the $y$-axis corresponds to $\log(t)$, where $t$ denotes the computation time (in seconds). We can observe from the figure that the computation time of both algorithms increases as $m$ increases and SOCP is faster than MMV$_{prox}$ on small problems ($m \leq 40$); when $m > 40$, MMV$_{prox}$ outperforms SOCP; when the value of $m$ is large ($m > 80$), the SOCP formulation cannot be solved by SeDuMi, while MMV$_{prox}$ can still be applied. This experimental result demonstrates the good scalability of the proposed MMV$_{prox}$ algorithm in comparison with the SOCP formulation.

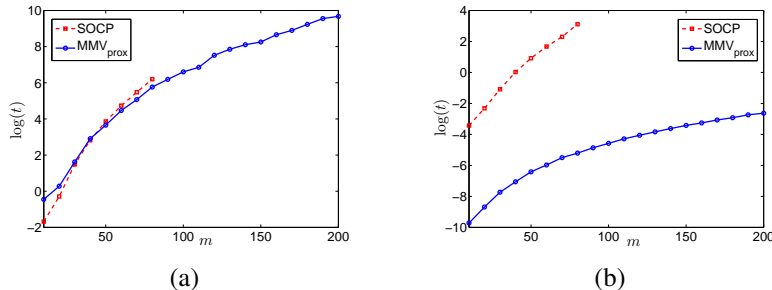

(a)  (b)

Figure 2: Scalability comparison of MMV$_{prox}$ and SOCP: (a) the computation time for both algorithms as the problem size varies; and (b) the average computation time of each iteration for both algorithms as the problem size varies. The $x$-axis denotes the value of $m$, and the $y$-axis denotes the computation time in seconds (in log scale).

To further examine the scalability of both algorithms, we compare the execution time of each iteration for both SOCP and the proposed algorithm. We use the same setting as in the last experiment, i.e., $n = 2m, d = 4m$, and $m$ ranges from 10 to 200 with a step size of 10. The time comparison of SOCP and MMV$_{prox}$ is presented in Figure 2 (b). We observe that MMV$_{prox}$ has a significantly lower cost than SOCP in each iteration (note that SOCP is not applicable for $m > 80$). This is consistent with our complexity analysis in Section 4.

We can observe from Figure 2 that when $m$ is small, the computation time of SOCP and MMV$_{prox}$ is comparable, although MMV$_{prox}$ is much faster in each iteration. This is because MMV$_{prox}$ is a first-order method, which has a slower convergence rate than the second-order method SOCP. Thus, there is a tradeoff between scalability and convergence rate. Our experiments show the advantage of MMV$_{prox}$ for large-size problems.

## 6 Conclusions

In this paper, we consider the $(2,1)$-norm minimization for the reconstruction of sparse signals in the multiple measurement vector (MMV) model, in which the signal consists of a set of jointly sparse vectors. Existing algorithms formulate it as second-order cone programming or semdefinite programming, which is computationally expensive to solve for problems of moderate size. In this paper, we propose an equivalent dual formulation for the $(2,1)$-norm minimization in the MMV model, and develop the MMV$_{prox}$ algorithm for solving the dual formulation based on the prox-method. In addition, our theoretical analysis reveals the close connection between the proposed dual formulation and multiple kernel learning. Our simulation studies demonstrate the effectiveness of the proposed algorithm in terms of recovery quality and scalability. In the future, we plan to compare existing solvers for multiple kernel learning [14, 22, 27] with the proposed MMV$_{prox}$ algorithm. In addition, we plan to examine the efficiency of the prox-method for solving various MKL formulations.

## Acknowledgements

This work was supported by NSF IIS-0612069, IIS-0812551, CCF-0811790, NIH R01-HG002516, NGA HM1582-08-1-0016, and NSFC 60905035.

# References

[1] A. Ben-Tal and A. Nemirovski. Non-Euclidean restricted memory level method for large-scale convex optimization. *Mathematical Programming*, 102(3):407–56, 2005.

[2] S. Boyd and L. Vandenberghe. *Convex Optimization*. Cambridge University Press, Cambridge, UK, 2004.

[3] E. Candès. Compressive sampling. In *International Congress of Mathematics*, number 3, pages 1433–1452, Madrid, Spain, 2006.

[4] E. Candès, J. Romberg, and T. Tao. Robust uncertainty principles: exact signal reconstruction from highly incomplete frequency information. *IEEE Transactions on Information Theory*, 52(2):489–509, 2006.

[5] J. Chen and X. Huo. Theoretical results on sparse representations of multiple-measurement vectors. *IEEE Transactions on Signal Processing*, 54(12):4634–4643, 2006.

[6] S.F. Cotter and B.D. Rao. Sparse channel estimation via matching pursuit with application to equalization. *IEEE Transactions on Communications*, 50(3):374–377, 2002.

[7] S.F. Cotter, B.D. Rao, Kjersti Engan, and K. Kreutz-Delgado. Sparse solutions to linear inverse problems with multiple measurement vectors. *IEEE Transactions on Signal Processing*, 53(7):2477–2488, 2005.

[8] D.L. Donoho. Compressed sensing. *IEEE Transactions on Information Theory*, 52(4):1289–1306, 2006.

[9] D.L. Duttweiler. Proportionate normalized least-mean-squares adaptation in echo cancelers. *IEEE Transactions on Speech and Audio Processing*, 8(5):508–518, 2000.

[10] Y.C. Eldar and M. Mishali. Robust recovery of signals from a structured union of subspaces. *To Appear in IEEE Transactions on Information Theory*, 2009.

[11] S. Erickson and C. Sabatti. Empirical bayes estimation of a sparse vector of gene expression changes. *Statistical Applications in Genetics and Molecular Biology*, 4(1):22, 2008.

[12] I.F. Gorodnitsky, J.S. George, and B.D. Rao. Neuromagnetic source imaging with focuss: a recursive weighted minimum norm algorithm. *Electroencephalography and Clinical Neurophysiology*, 95(4):231–251, 1995.

[13] H. Jean-Baptiste and C. Lemarechal. *Convex Analysis and Minimization Algorithms I: Fundamentals (Grundlehren Der Mathematischen Wissenschaften)*. Springer, Berlin, 1993.

[14] G.R.G. Lanckriet, N. Cristianini, P. Bartlett, L. El Ghaoui, and M.I. Jordan. Learning the kernel matrix with semidefinite programming. *Jouranl of Machine Learning Research*, 5:27–72, 2004.

[15] M. Lobo, L. Vandenberghe, S. Boyd, and H. Lebret. Applications of second-order cone programming. *Linear Algebra and its Applications*, 284(1-3):193–228, 1998.

[16] F. Parvaresh M. Stojnic and B. Hassibi. On the reconstruction of block-sparse signals with an optimal number of measurements. *CoRR*, 2008.

[17] D. Malioutov, M. Cetin, and A. Willsky. Source localization by enforcing sparsity through a laplacian. In *IEEE Workshop on Statistical Signal Processing*, pages 553–556, 2003.

[18] M. Mishali and Y.C. Eldar. Reduce and boost: Recovering arbitrary sets of jointly sparse vectors. *IEEE Transactions on Signal Processing*, 56(10):4692–4702, 2008.

[19] A. Nemirovski. Prox-method with rate of convergence $o(1/t)$ for variational inequalities with Lipschitz continuous monotone operators and smooth convex-concave saddle point problems. *SIAM Journal on Optimization*, 15(1):229–251, 2005.

[20] Y.E. Nesterov and A.S. Nemirovskii. *Interior-point Polynomial Algorithms in Convex Programming*. SIAM Publications, Philadelphia, PA, 1994.

[21] M. Duarte R.G. Baraniuk, V. Cevher and C. Hegde. Model-based compressive sensing. *Submitted to IEEE Transactions on Information Theory*, 2008.

[22] S. Sonnenburg, G. Rätsch, C. Schäfer, and B. Schölkopf. Large scale multiple kernel learning. *Journal of Machine Learning Research*, 7:1531–1565, 2006.

[23] J.F. Sturm. Using sedumi 1.02, a MATLAB toolbox for optimization over symmetric cones. *Optimization Methods and Software*, 11(12):625–653, 1999.

[24] J.A. Tropp. Algorithms for simultaneous sparse approximation. Part II: Convex relaxation. *Signal Processing*, 86(3):589–602, 2006.

[25] J.A. Tropp, A.C. Gilbert, and M.J. Strauss. Algorithms for simultaneous sparse approximation. Part I: Greedy pursuit. *Signal Processing*, 86(3):572–588, 2006.

[26] E. van den Berg and M. P. Friedlander. Joint-sparse recovery from multiple measurements. *Technical Report, Department of Computer Science, University of British Columbia*, 2009.

[27] Z. Xu, R. Jin, I. King, and M.R. Lyu. An extended level method for efficient multiple kernel learning. In *Advances in Neural Information Processing Systems*, pages 1825–1832, 2008.

